# Learning a Rare Event Detection Cascade by Direct Feature Selection

**Jianxin Wu     James M. Rehg     Matthew D. Mullin**
College of Computing and GVU Center, Georgia Institute of Technology
{wujx, rehg, mdmullin}@cc.gatech.edu

## Abstract

Face detection is a canonical example of a rare event detection problem, in which target patterns occur with much lower frequency than non-targets. Out of millions of face-sized windows in an input image, for example, only a few will typically contain a face. Viola and Jones recently proposed a cascade architecture for face detection which successfully addresses the rare event nature of the task. A central part of their method is a feature selection algorithm based on AdaBoost. We present a novel cascade learning algorithm based on forward feature selection which is two orders of magnitude faster than the Viola-Jones approach and yields classifiers of equivalent quality. This faster method could be used for more demanding classification tasks, such as on-line learning.

## 1 Introduction

Fast and robust face detection is an important computer vision problem with applications to surveillance, multimedia processing, and HCI. Face detection is often formulated as a search and classification problem: a search strategy generates potential image regions and a classifier determines whether or not they contain a face. A standard approach is brute-force search, in which the image is scanned in raster order and every $n \times n$ window of pixels over multiple image scales is classified [1, 2, 3].

When a brute-force search strategy is used, face detection is a *rare event detection* problem, in the sense that among the millions of image regions, only very few contain faces. The resulting classifier design problem is very challenging: The detection rate must be very high in order to avoid missing any rare events. At the same time, the false positive rate must be very low (e.g. $10^{-6}$) in order to dodge the flood of non-events. From the computational standpoint, huge speed-ups are possible if the sparsity of faces in the input set can be exploited. In their seminal work [4], Viola and Jones proposed a face detection method based on a cascade of classifiers, illustrated in figure 1. Each classifier node is designed to reject a portion of the nonface regions and pass all of the faces. Most image regions are rejected quickly, resulting in very fast face detection performance.

There are three elements in the Viola-Jones framework: the cascade architecture, a rich over-complete set of rectangle features, and an algorithm based on AdaBoost for constructing ensembles of rectangle features in each classifier node. Much of the recent work on face detection following Viola-Jones has explored alternative boosting algorithms such as Float-Boost [5], GentleBoost [6], and Asymmetric AdaBoost [7] (see [8] for a related method).

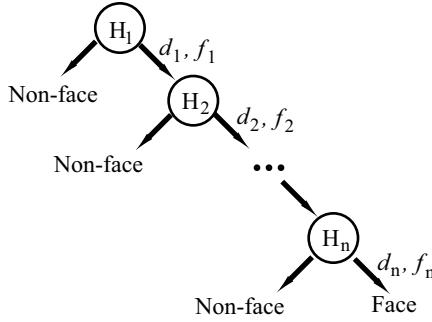

Figure 1: Illustration of the cascade architecture with $n$ nodes.

This paper is motivated by the observation that the AdaBoost feature selection method is an *indirect* way to meet the learning goals of the cascade. It is also an expensive algorithm. For example, weeks of computation are required to produce the final cascade in [4].

In this paper we present a new cascade learning algorithm which uses direct forward feature selection to construct the ensemble classifiers in each node of the cascade. We demonstrate empirically that our algorithm is two orders of magnitude faster than the Viola-Jones algorithm, and produces cascades which are indistinguishable in face detection performance. This faster method could be used for more demanding classification tasks, such as on-line learning or searching the space of classifier structures. Our results also suggest that a large portion of the effectiveness of the Viola-Jones detector should be attributed to the cascade design and the choice of the feature set.

## 2 Cascade Architecture for Rare Event Detection

The learning goal for the cascade in figure 1 is the construction of a set of classifiers $\{H_i\}_{i=1}^{n}$. Each $H_i$ is required to have a very high detection rate, but only a *moderate* false positive rate (e.g. 50%). An input image region is passed from $H_i$ to $H_{i+1}$ if it is classified as a face, otherwise it is rejected. If the $\{H_i\}$ can be constructed to produce *independent* errors, then the overall detection rate $d$ and false positive rate $f$ for the cascade is given by $\prod_{i=1}^{n} d_i$ and $\prod_{i=1}^{n} f_i$ respectively. In a hypothetical example, a 20 node cascade with $d_i = 0.999$ and $f_i = 0.5$ would have $d = 0.98$ and $f = 9.6e - 7$.

As in [4], the overall cascade learning method in this paper is a stage-wise, greedy feature selection process. Nodes are constructed sequentially, starting with $H_1$. Within a node $H_i$, features are added sequentially to form an ensemble. Following Viola-Jones, the training dataset is manipulated between nodes to encourage independent errors. Each node $H_i$ is trained on all of the positive examples and a subset of the negative examples. In moving from node $H_i$ to $H_{i+1}$ during training, negative examples that were classified successfully by the cascade are discarded and replaced with new ones, using the standard bootstrapping approach from [1]. The difference between our method and Viola-Jones is the feature selection algorithm for the individual nodes.

The cascade architecture in figure 1 should be suitable for other rare event problems, such as network intrusion detection in which an attack constitutes a few packets out of tens of millions. Recent work in that community has also explored a cascade approach [9].

For each node in the cascade architecture, given a training set $\{x_i, y_i\}$, the learning objective is to select a set of weak classifiers $\{h_t\}$ from a total set of $F$ features and combine them into an ensemble $H$ with a high detection rate $d$ and a moderate false positive rate $f$.

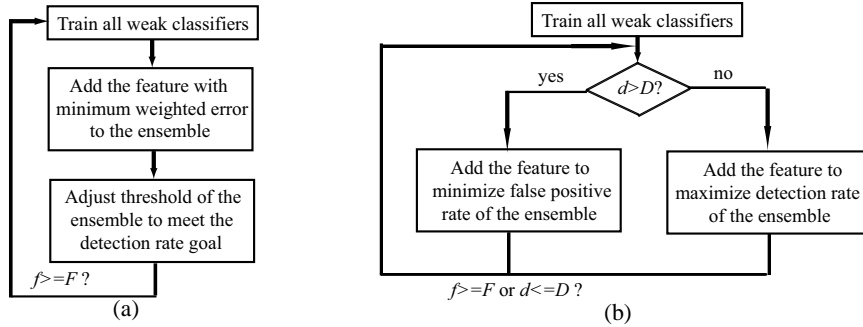

Figure 2: Diagram for training one node in the cascade architecture, (a) is for the Viola-Jones method, and (b) is for the proposed method. *F* and *D* are false positive rate and detection rate goals respectively.

A weak classifier is formed from a rectangle feature by applying the feature to the input pattern and thresholding the result.[1] Training a weak classifier corresponds to setting its threshold.

In [4], an algorithm based on AdaBoost trains weak classifiers, adds them to the ensemble, and computes the ensemble weights. AdaBoost [10] is an iterative method for obtaining an ensemble of weak classifiers by evolving a distribution of weights, $D_t$, over the training data. In the Viola-Jones approach, each iteration $t$ of boosting adds the classifier $h_t$ with the lowest weighted error to the ensemble. After $T$ rounds of boosting, the decision of the ensemble is defined as $H(x) = \begin{cases} 1 & \sum_{t=1}^{T} \alpha_t h_t(x) \geq \theta \\ 0 & \text{otherwise} \end{cases}$ , where the $\alpha_t$ are the standard AdaBoost ensemble weights and $\theta$ is the threshold of the ensemble. This threshold is adjusted to meet the detection rate goal. More features are then added if necessary to meet the false positive rate goal. The flowchart for the algorithm is given in figure 2(a).

The process of sequentially adding features which individually minimize the weighted error is at best an indirect way to meet the learning goals for the ensemble. For example, the false positive goal is relatively easy to meet, compared to the detection rate goal which is near 100%. As a consequence, the threshold $\theta$ produced by AdaBoost must be discarded in favor of a threshold computed directly from the ensemble performance. Unfortunately, the weight distribution maintained by AdaBoost requires that the complete set of weak classifiers be retrained in each iteration. This is a computationally demanding task which is in the inner loop of the feature selection algorithm.

Beyond these concerns is a more basic question about the cascade learning problem: *What is the role of boosting in forming an effective ensemble?* Our hypothesis is that the overall success of the method depends upon having a sufficiently rich feature set, which defines the space of possible weak classifiers. From this perspective, a failure mode of the algorithm would be the inability to find sufficient features to meet the learning goal. The question then is to what extent boosting helps to avoid this problem. In the following section we describe a simple, direct feature selection algorithm that sheds some light on these issues.

## 3   Direct Feature Selection Method

We propose a new cascade learning algorithm based on forward feature selection [11]. Pseudo-code of the algorithm for building an ensemble classifier for a single node is given

1. Given a training set. Given $d$, the minimum detection rate and $f$, the maximum false positive rate.

2. For every feature, $j$, train a weak classifier $h_j$, whose false positive rate is $f$.

3. Initialize the ensemble H to an empty set, i.e. $H \leftarrow \phi$. $t \leftarrow 0$, $d_0 = 0.0$, $f_0 = 1.0$.

4. while $d_t < d$ or $f_t > f$

    (a) if $d_t < d$, then, find the feature $k$, such that by adding it to $H$, the new ensemble will have largest detection rate $d_{t+1}$.

    (b) else, find the feature $k$, such that by adding it to $H$, the new ensemble will have smallest false positive rate $f_{t+1}$.

    (c) $t \leftarrow t + 1$, $H \leftarrow H \cup \{h_k\}$.

5. The decision of the ensemble classifier is formed by a majority voting of weak classifiers in $H$, i.e. $H(x) = \begin{cases} 1 & \sum_{h_j \in H} h_j(x) \geq \theta \\ 0 & \text{otherwise} \end{cases}$ , where $\theta = \frac{T}{2}$. Decrease $\theta$ if necessary.

Table 1: The direct feature selection method for building an ensemble classifier.

in table 1. The corresponding flowchart is illustrated in figure 2(b). The first step in our algorithm is to train each of the weak classifiers to meet the false positive rate goal for the ensemble.

The output of each weak classifier on each training data item is collected in a large look-up table. The core algorithm is an exhaustive search over possible classifiers. In each iteration, we consider adding each possible classifier to the ensemble and select the one which makes the largest improvement to the ensemble performance. The selection criteria directly maximizes the learning objective for the node. The look-up table, in conjunction with majority vote rule, makes this feature search extremely fast.

The resulting algorithm is roughly 100 times faster than Viola-Jones. The key difference is that we train the weak classifiers only once per node, while in the Viola-Jones method they are trained once for each feature in the cascade. Let $T$ be the training time for weak classifiers[2] and $F$ be the number of features in the final cascade. The learning time for Viola-Jones is roughly $FT$, which in [4] was on the order of weeks. Let $N$ be the number of nodes in the cascade. Empirically the learning time for our method is $2NT$, which is on the order of hours in our experiments. For the cascade of 32 nodes with 4297 features in [4], the difference in learning time will be dramatic.

The difficulty of the classifier design problem increases with the depth of the cascade, as the non-face patterns selected by bootstrapping become more challenging. A large number of features may be required to achieve the learning objectives when majority vote is used. In this case, a weighted ensemble could be advantageous. Once feature selection has been performed, a variant of the Viola-Jones algorithm can be used to obtain a weighted ensemble. Pseudo-code for this weight setting method is given in table 2.

## 4  Experimental Results

We conducted three controlled experiments to compare our feature selection method to the Viola-Jones algorithm. The procedures and data sets were the same for all of the ex-

1. Given a training set, maintain a distribution $D$ over it.

2. Select $N$ features using the algorithm in table 1. These features form a set $F$.

3. Initialize the ensemble classifier to an empty set, i.e. $H \leftarrow \emptyset$.

4. for $i = 1 : N$

   (a) Select the feature $k$ from $F$ that has smallest error $\epsilon$ on the training set, weighted over the distribution $D$.

   (b) Update the distribution $D$ according to the AdaBoost algorithm as in [4].

   (c) Add the feature $k$ and it's associated weight $\alpha_k = -\log \frac{\epsilon}{1-\epsilon}$ to $H$. And remove the feature $k$ from $F$.

5. Decision of the ensemble classifier is formed by a weighted average of weak classifiers in $H$. Decrease the threshold $\theta$ until the ensemble reaches the detection rate goal.

Table 2: Weight setting algorithm after feature selection.

periments. Our training set contained 5000 example face images and 5000 initial non-face examples, all of size 24x24. We used approximately 2284 million non-face patches to bootstrap the non-face examples between nodes. We used 32466 features sampled uniformly from the entire set of rectangle features. For testing purposes we used the MIT+CMU frontal face test set [2] in all experiments. Although many researchers use automatic procedures to evaluate their algorithm, we decided to manually count the missed faces and false positives.[3] When scanning a test image at different scales, the image is re-scaled repeatedly by a factor of 1.25. Post-processing is similar to [4].

In the first experiment we constructed three face detection cascades. One cascade used the direct feature selection method from table 1. The second cascade used the weight setting algorithm in table 2. The training algorithms stopped when they exhausted the set of non-face training examples. The third cascade used our implementation of the Viola-Jones algorithm. The three cascades had 38, 37, and 28 nodes respectively. The third cascade was stopped after 28 nodes because the AdaBoost based training algorithm could not meet the learning goal. With 200 features, when the detection rate is 99.9%, the AdaBoost ensemble's false positive rate is larger than 97%. Adding several hundred additional features did not change the outcome. ROC curves for cascades using our method and the Viola-Jones method are depicted in figure 3(a). We constructed the ROC curves by removing nodes from the cascade to generate points with increasing detection and false positive rates. These curves demonstrate that the test performance of our method is indistinguishable from that of the Viola-Jones method.

The second experiment explored the ability of the rectangle feature set to meet the detection rate goal for the ensemble on a difficult node. Figure 3(b) shows the false positive and detection rates for the ensemble (i.e., one node in the cascade architecture) as a function of the number of features that were added to the ensemble. The training set used was the bootstrapped training set for the 19th node in the cascade which was trained by the Viola-Jones method. Even for this difficult learning task, the algorithm can improve the detection rate from about 0.7 to 0.9 using only 13 features, without any significant increase in false positive rate. This suggests that the rectangle feature set is sufficiently rich. Our hypothesis is that the strength of this feature set in the context of the cascade architecture is the key to

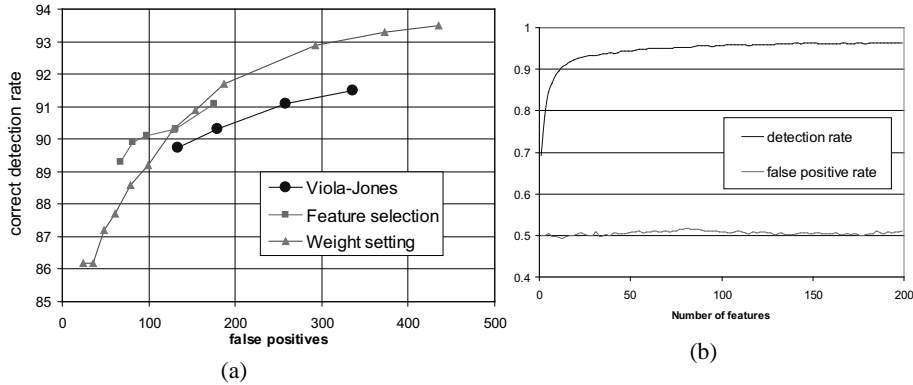

(a)

(b)

Figure 3: Experimental Results. (a) is ROC curves of the proposed method and the Viola-Jones method and (b) is trend of detection and false positive rates when more features are combined in one node.

the success of the Viola-Jones approach.

We conducted a third experiment in which we focused on learning one node in the cascade architecture. Figure 4 shows ROC curves of the Viola-Jones, direct feature selection, and weight setting methods for one node of the cascade. The training set used in figure 4 was the same training set as in the second experiment. Unlike the ROC curves in figure 3(a), these curves show the performance of the node in isolation using a validation set. These curves reinforce the similarity in the performance of our method compared to Viola-Jones. In the region of interest (e.g. detection rate $> 99\%$), our algorithms yield better ROC curve performance than the Viola-Jones method. Although figure 4 and figure 3(b) only showed curves for one specific training set, the same pattern in these figures were found with other bootstrapped training sets in our experiments.

## 5 Related Work

A survey of face detection methods can be found in [12]. We restrict our attention here to frontal face detection algorithms related to the cascade idea. The neural network-based detector of Rowley et. al. [2] incorporated a manually-designed two node cascade. Other cascade structures have been constructed for SVM classifiers. In [13], a set of reduced set vectors is calculated from the support vectors. Each reduced set vector can be interpreted as a face or anti-face template. Since these reduced set vectors are applied *sequentially* to the input pattern, they can be viewed as nodes in a cascade. An alternative cascade framework for SVM classifiers is proposed by Heisele et. al. in [14]. Based on different assumptions, Keren et al. proposed another object detection method which consists of a series of anti-face templates [15]. Carmichael and Hebert propose a hierarchical strategy for detecting chairs at different orientations and scales [16].

Following [4], several authors have developed alternative boosting algorithms for feature selection. Li et al. incorporated floating search into the AdaBoost algorithm (FloatBoost) and proposed some new features for detecting multi-view faces [5]. Lienhart et al. [6] experimentally evaluated different boosting algorithms and different weak classifiers. Their results showed that Gentle AdaBoost and CART decision trees had the best performance. In an extension of their original work [7], Viola and Jones proposed an asymmetric AdaBoost algorithm in which false negatives are penalized more than false positives. This is an interesting attempt to incorporate the rare event observation more explicitly into their

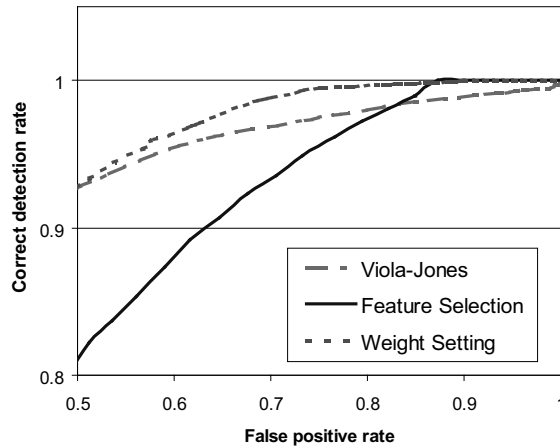

Figure 4: Single node ROC curves on a validation set.

learning algorithm (see [8] for a related method). All of these methods explore variations in AdaBoost-based feature selection, and their training times are similar to the original Viola-Jones algorithm. While all of the above methods adopt a brute-force search strategy for generating input regions, there has been some interesting work on generating candidate face hypotheses from more general interest operators. Two examples are [17, 18].

## 6    Conclusions

Face detection is a canonical example of a rare event detection task, in which target patterns occur with much lower frequency than non-targets. It results in a challenging classifier design problem: The detection rate must be very high in order to avoid missing any rare events and the false positive rate must be very low to dodge the flood of non-events. A cascade classifier architecture is well-suited to rare event detection.

The Viola-Jones face detection framework consists of a cascade architecture, a rich over-complete feature set, and a learning algorithm based on AdaBoost. We have demonstrated that a simpler direct algorithm based on forward feature selection can produce cascades of similar quality with two orders of magnitude less computation. Our algorithm directly optimizes the learning criteria for the ensemble, while the AdaBoost-based method is more indirect. This is because the learning goal is a highly-skewed tradeoff between detection rate and false positive rate which does not fit naturally into the weighted error framework of AdaBoost. Our experiments suggest that the feature set and cascade structure in the Viola-Jones framework are the key elements in the success of the method.

Three issues that we plan to explore in future work are: the necessary properties for feature sets, global feature selection methods, and the incorporation of search into the cascade framework. The rectangle feature set seems particularly well-suited for face detection. What general properties must a feature set possess to be successful in the cascade framework? In other rare event detection tasks where a large set of diverse features is not naturally available, methods to create such a feature set may be useful (e.g. the random subspace method proposed by Ho [19]).

In our current algorithm, both nodes and features are added sequentially and greedily to the cascade. More global techniques for forming ensembles could yield better results.

Finally, the current detection method relies on a brute-force search strategy for generating candidate regions. We plan to explore the cascade architecture in conjunction with more general interest operators, such as those defined in [18, 20].

The authors are grateful to Mike Jones and Paul Viola for providing their training data, along with many valuable discussions. This work was supported by NSF grant IIS-0133779 and the Mitsubishi Electric Research Laboratory.

## Footnotes

[1]A feature and its corresponding classifier will be used interchangeably.

[2]In our experiments, $T$ is about 10 minutes.

[3]We found that the criterion for automatically finding detection errors in [6] was too loose. This criterion yielded higher detection rates and lower false positive rates than manual counting.

## References

[1] K. Sung and T. Poggio. Example-based learning for view-based human face detection. *IEEE Trans. on Pattern Analysis and Machine Intelligence*, 20(1):39–51, 1998.

[2] H. A. Rowley, S. Baluja, and T. Kanade. Neural network-based face detection. *IEEE Trans. on Pattern Analysis and Machine Intelligence*, 20(1):23–38, 1998.

[3] Henry Schneiderman and Takeo Kanade. A statistical model for 3d object detection applied to faces and cars. In *IEEE Conference on Computer Vision and Pattern Recognition*. IEEE, June 2000.

[4] P. Viola and M. Jones. Rapid object detection using a boosted cascade of simple features. In *Proc. CVPR*, pages 511–518, 2001.

[5] S.Z. Li, Z.Q. Zhang, Harry Shum, and H.J. Zhang. FloatBoost learning for classification. In S. Thrun S. Becker and K. Obermayer, editors, *NIPS 15*. MIT Press, December 2002.

[6] R. Lienhart, A. Kuranov, and V. Pisarevsky. Empirical analysis of detection cascades of boosted classifiers for rapid object detection. Technical report, MRL, Intel Labs, 2002.

[7] P. Viola and M. Jones. Fast and robust classification using asymmetric AdaBoost and a detector cascade. In *NIPS 14*, 2002.

[8] G. J. Karakoulas and J. Shawe-Taylor. Optimizing classifiers for imbalanced training sets. In *NIPS 11*, pages 253–259, 1999.

[9] W. Fan, W. Lee, S. J. Stolfo, and M. Miller. A multiple model cost-sensitive approach for intrusion detection. In *Proc. 11th ECML*, 2000.

[10] R. E. Schapire, Y. Freund, P. Bartlett, and W. S. Lee. Boosting the margin: A new explanation for the effectiveness of voting methods. *The Annals of Statistics*, 26(5):1651–1686, 1998.

[11] A. R. Webb. *Statistical Pattern Recognition*. Oxford University Press, New York, 1999.

[12] M.-H. Yang, D. J. Kriegman, and N. Ahujua. Detecting faces in images: a survey. *IEEE Trans. on Pattern Analysis and Machine Intelligence*, 24(1):34–58, 2002.

[13] S. Romdhani, P. Torr, B. Schoelkopf, and A. Blake. Computationally efficient face detection. In *Proc. Intl. Conf. Computer Vision*, pages 695–700, 2001.

[14] B. Heisele, T. Serre, S. Mukherjee, and T. Poggio. Feature reduction and hierarchy of classifiers for fast object detection in video images. In *Proc. CVPR*, volume 2, pages 18–24, 2001.

[15] D. Keren, M. Osadchy, and C. Gotsman. Antifaces: A novel, fast method for image detection. *IEEE Trans. on Pattern Analysis and Machine Intelligence*, 23(7):747–761, 2001.

[16] O. Carmichael and M. Hebert. Object recognition by a cascade of edge probes. In *British Machine Vision Conference*, volume 1, pages 103–112, September 2002.

[17] T. Leung, M. Burl, and P. Perona. Finding faces in cluttered scenes using random labeled graph matching. In *Proc. Intl. Conf. Computer Vision*, pages 637–644, 1995.

[18] S. Lazebnik, C. Schmid, and J. Ponce. Sparse texture representation using affine-invariant neighborhoods. In *Proc. CVPR*, 2003.

[19] T. K. Ho. The random subspace method for constructing decision forests. *IEEE Trans. on Pattern Analysis and Machine Intelligence*, 20(8):832–844, 1998.

[20] S. Belongie, J. Malik, and J. Puzicha. Shape matching and object recognition using shape contexts. *IEEE Trans. on Pattern Analysis and Machine Intelligence*, 24(4):509–522, 2002.
